# Exploring Functional Connectivity of the Human Brain using Multivariate Information Analysis

**Barry Chai**[1*]  **Dirk B. Walther**[2*]  **Diane M. Beck**[2,3†]  **Li Fei-Fei**[1†]

[1]Computer Science Department, Stanford University, Stanford, CA 94305
[2]Beckman Institute, University of Illinois at Urbana-Champaign, Urbana, IL 61801
[3]Psychology Department, University of Illinois at Urbana-Champaign, Champaign, IL 61820
{bwchai,feifeili}@cs.stanford.edu  {walther,dmbeck}@illinois.edu

## Abstract

In this study, we present a new method for establishing fMRI pattern-based functional connectivity between brain regions by estimating their multivariate mutual information. Recent advances in the numerical approximation of high-dimensional probability distributions allow us to successfully estimate mutual information from scarce fMRI data. We also show that selecting voxels based on the multivariate mutual information of local activity patterns with respect to ground truth labels leads to higher decoding accuracy than established voxel selection methods. We validate our approach with a 6-way scene categorization fMRI experiment. Multivariate information analysis is able to find strong information sharing between PPA and RSC, consistent with existing neuroscience studies on scenes. Furthermore, an exploratory whole-brain analysis uncovered other brain regions that share information with the PPA-RSC scene network.

## 1 Introduction

To understand how the brain represents and processes information we must account for two complementary properties: information is represented in a distributed fashion, and brain regions are strongly interconnected. Although heralded as a tool to address these issues, functional magnetic resonance imaging (fMRI) initially fell short of achieving these goals because of limitations of traditional analysis methods, which treat voxels as independent. Multi-voxel pattern analysis (MVPA) has revolutionized fMRI analysis by accounting for distributed patterns of activity rather than absolute activation levels. The analysis of functional connectivity, however, is so far mostly limited to comparing the time courses of individual voxels. To overcome these limitations we demonstrate a new method of pattern-based functional connectivity analysis based on mutual information of sets of voxels. Furthermore, we show that selecting voxels based on the mutual information of local activity with respect to ground truth outperforms other voxel selection methods.

We apply our new analysis methods to the decoding of natural scene categories from the human brain. Human observers are able to quickly and efficiently perceive the content of natural scenes [15, 26]. It was recently shown by [23] that activity patterns in the parahippocampal place area (PPA), the retrosplenial cortex (RSC), the lateral occipital complex (LOC), and, to some degree, primary visual cortex (V1) contain information about the categories of natural scenes. To truly understand how the brain categorizes natural scenes, however, it is necessary to grasp the interactions between these regions of interest (ROIs). Our new technique for pattern-based functional connectivity enables us to uncover shared scene category-specific information among the ROIs. When configured for exploratory whole-brain analysis, the technique allows us to discover other brain regions that may be involved in natural scene categorization.

Mutual information is appropriate for fMRI analysis if one considers fMRI data as a noisy communication channel in the sense of Shannon's information theory [19]; the information contained

in a population of neurons must be communicated through hemodynamic changes and concomitant changes in magnetization which can be measured as the blood-oxygen level dependent (BOLD) fMRI signal, then proceed through several layers of data processing, culminating in a single time varying value in a particular voxel. While this noisy communication concept has been embraced by the brain machine interface community [25], information theory has, thus far, been less utilized in the fMRI analysis community (see [8] for exceptions). This may be partly due to the numerical difficulties in estimating the probability distributions necessary for computing mutual information. This problem is exacerbated when patterns of voxels are considered. In this case distributions of higher dimensionality need to be estimated from preciously few data points. Recent developments in information theory, however, help us overcome these hurdles.

In Section 2 we review these theoretical advances and adapt them for our dual purpose of voxel selection and pattern-based functional connectivity analysis. Following a discussion of related work in Section 3, in Section 4 we apply our new methods to fMRI data from an experiment on distinguishing natural scene categories in the human brain. We lay conclude the paper in Section 5.

## 2 Multivariate mutual information for fMRI data

Information theory was originally formulated for discrete variables. In order to adapt the theory to continuous random variables, the underlying probability distribution needs to be estimated from the sampled data points. Previous work such as [7, 18] have used fixed bin-size histogram or Parzen window methods for this purpose. However, these methods do not generalize to high-dimensional data. Recently, Perez-Cruz has shown that a k-nearest-neighbor (kNN) approach to estimating information theoretic measures converges to the true information theoretic measures asymptotically with finite k, even in higher dimensional spaces [16]. In this section we adapt this strategy to estimate multi-voxel mutual information.

### 2.1 Nearest-neighbor mutual information estimate

In information theory, the randomness of a probability distribution is measured by its entropy. For a discrete random variable $x$, entropy can be calculated as

$$H(x) = -\sum_{i=1}^{n} p(x_i) \log p(x_i). \tag{1}$$

Mutual information is intuitively defined as the reduction of the entropy of the random variable $x$ by the entropy of $x$ after $y$ is known:

$$I(x, y) = H(x) - H(x|y). \tag{2}$$

The separation into entropies allow us to calculate mutual information for multivariate data. Random variables $x$, $y$ can be of arbitrary dimensions.

As shown in [24], using kNN estimation, entropies and conditional entropies can be defined as

$$H(x) = -\frac{1}{n} \sum_{i=1}^{n} \log p_k(x_i), \tag{3}$$

$$H(x|y) = -\frac{1}{n} \sum_{i=1}^{n} \log \frac{p_k(x_i, y_i)}{p_k(y_i)}, \tag{4}$$

where the summation is over $n$ data points, each represented by $x_i$. $p_k(x_i)$ is the kNN density estimated at $x_i$. $p_k(x_i)$ is defined as

$$p_k(x_i) = \frac{k}{n-1} \frac{\Gamma(d/2 + 1)}{\pi^{d/2}} \frac{1}{r_k(x_i)^d}. \tag{5}$$

where $\Gamma$ is the gamma function, $d$ is the dimensionality of $x_i$ and $r_k(x_i)$ is the Euclidean distance from $x_i$ to the $k^{th}$ nearest training point. $p_k(x_i)$ is the probability density function at $x_i$, which is a set of voxel values for a given category task(or label) in the context of our fMRI experiment.

## 2.2 fMRI multivariate information analysis

In previous work, such as [7], information theory has been used as a measure for functional connectivity of one voxel to another voxel. While such analysis is valuable for exploring connections in the brain, it does not fully leverage the information stored in the local pattern of voxels. In this section we propose a framework for multivariate information analysis of fMRI data for dual purposes: voxel selection and functional connectivity.

### 2.2.1 Voxel selection based on mutual information with respect to ground truth label

For voxel selection we are interested in finding a subset of voxels that are highly informative for discriminating between the ground truth labels in the experiment. This is a useful step that serves two purposes. From a machine learning perspective, reducing the dimensionality of the brain image data can boost classifier performance and reduce classifier variance. From a neuroscience perspective, the locations of highly informative voxels identify functional regions involved in the experiment. To achieve both of these goals we use a multivariate mutual information measure to analyze a localized pattern of M voxels. This local analysis windows is moved across the brain image. At each location we estimate the mutual information shared between the pattern of M voxels and the experiment label. In our experiments we choose M = 7 to evaluate the smallest symmetrical pattern around a center voxel, which consists of the center voxel and its 6 face-connected neighbors. Mutual information between voxels $V$ and labels $L$ is defined as

$$I(V, L) = H(V) + H(L) - H(V, L). \tag{6}$$

Using equation 1 the entropies can be calculated by

$$I(V, L) = -\frac{1}{n} \sum_{i=1}^{n} \log p_k(V_i) - \frac{1}{n} \sum_{i=1}^{n} \log p_k(L_i) + \frac{1}{n} \sum_{i=1}^{n} \log p_k(V_i, L_i), \tag{7}$$

where $n$ is the number of data-points observed, $L_i$ is the experiment label for $i^{th}$ data point, $V_i$ is a 7-dimensional random variable, $V_i = (v_{i1}, v_{i2}, v_{i3}, v_{i4}, v_{i5}, v_{i6}, v_{i7})$ with each entry corresponding to one of 7 voxels' values at data point $i$. Equation 7 can be used to compute the mutual information of localized set of voxels $V_i$ with respect to their ground truth label $L_i$. We can then perform voxel selection by selecting the locations of highest mutual information. This is useful as a preprocessing step before applying any machine learning algorithms and as well as a way to spatially map out the informative voxels with respect to the task.

### 2.2.2 Functional connectivity by shared information between distributed voxel patterns

Two distributed brain regions can be modeled as a communication channel. Measuring the mutual information across the two regions provides an intuitive measure for their functional connectivity. The voxel values observed in each region can be regarded as observed data from an underlying probability distribution – the distribution that characterizes the functional region under the experiment condition.

Previous approaches have analyzed shared information in a univariate way, computing the mutual information between two voxels. However such univariate information analysis disregards the information stored in the local patterns of voxels. In this work we present a multivariate information analysis that estimates shared information between two sets of voxels that leverages the information stored in the local patterns:

$$I(V, S|L) = H(V|L) + H(S|L) - H(V, S|L), \tag{8}$$

where $V$ and $S$ are random variables for sets of 7 voxels. $L$ is the experiment label. Using equations 3 and 4 this can be written as

$$I(V, S|L) = -\frac{1}{n} \sum_{i=1}^{n} \log \frac{p_k(V_i, L_i)}{p_k(L_i)} - \frac{1}{n} \sum_{i=1}^{n} \log \frac{p_k(S_i, L_i)}{p_k(L_i)} + \frac{1}{n} \sum_{i=1}^{n} \log \frac{p_k(V_i, S_i, L_i)}{p_k(L_i)}. \tag{9}$$

Equation 9 allows us to measure the functional connectivity between two distributed sets of voxels $V$ and $S$ by computing the mutual information between the two sets of voxels conditioned on the experiment task label $L$. We show in our experiments (sec 4.4) that by using this measurement, our algorithm can uncover meaningful functional connectivity patterns among regions of the brain.

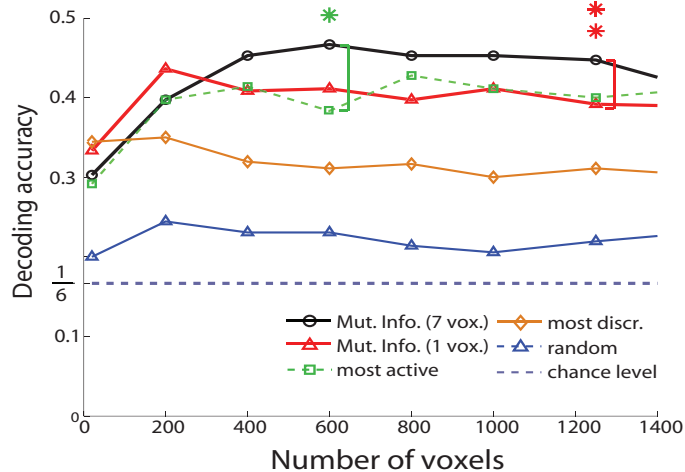

Figure 1: Comparison of decoding accuracy[3] between MI voxel selection and other standard voxel selection methods(refer to section 4.2). The single voxel MI approach surpasses *most discr.*[1] voxel selection but performs on par with *most active*[2] voxel selection. Using a pattern of 7 voxels, the MI7D approach achieves the highest decoding accuracy. At 600 voxels, MI7D decoding accuracy[3] is significantly higher than *most active* with p-value $< 0.05$. At 1250 voxels, MI7D decoding accuracy is significantly higher than MI1D with p-value $< 0.01$ (This figure must be viewed in color)

## 3  Related work

Statistical relationships between different parts of the brain, referred to as functional connectivity, have been computed with a number of different methods. The methods can be broadly classified as either data-driven or model-based [10].

In data-driven approaches, no specific hypothesis of connectivity is used, but large networks of brain regions are discovered based purely on the data. Most commonly, this is achieved with a dimensionality-reduction procedure such as principal component analysis (PCA) or independent component analysis (ICA). Originally applied to the analysis of PET data [5], PCA has also been applied to fMRI data (see [12]). ICA has been gained interest for the investigation of the so-called default network in the brain at rest [11].

Model-based approaches test a prior hypothesis about the statistical relations between a seed voxel and a target voxel. By fixing the seed voxel and moving the target voxel all over the brain, a connectivity map with respect to the seed voxel can be generated. The statistical relation of the two voxels is usually modeled assuming temporal dependence between voxels in methods such as: cross-correlation [2], coherence [21], Granger causality [1], or transfer entropy [20].

These methods compare the time courses of individual voxels. Following the same principal idea, we model functional connectivity based on the mutual information between sets of seed and target voxels to leverage the spatial information contained in activity patterns among voxels rather than the temporal information between two voxels. fMRI has a higher spatial resolution than temporal resolution. We design our mutual information connectivity measure to exploit this property of fMRI data. Yao et al. [26] have also explored pattern-based functional connectivity by modeling the interactions between distributed sets of voxels with a generative model. We take a simpler approach by using only the multivariate information measure which allows us to explore for unknown connections in the whole brain in a searchlight manner.

In recent years it has become apparent that patterns of fMRI activity hold more detailed information about experimental conditions than the activation levels of individual voxels [6]. It is therefore

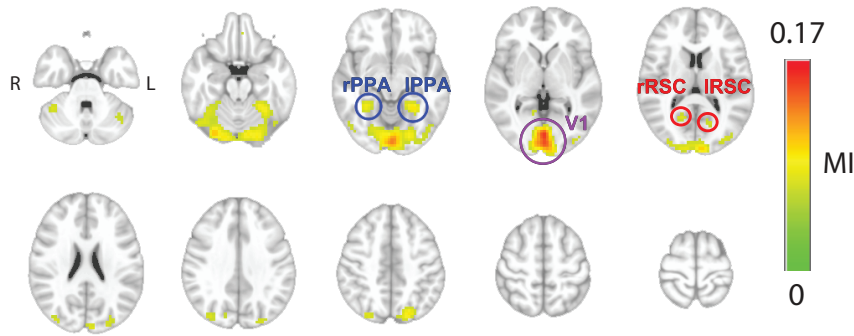

Figure 2: Locations of voxels with high 7D mutual information with respect to scene category label. The known functional areas that respond to scenes and visual stimuli such as PPA, RSC, V1 are all selected, which also explains the high decoding accuracy using the selected voxels. The brain maps shown above are based on group analysis over 5 subjects superimposed on an MNI standard brain. (This figure must be viewed in color)

cogent to also consider the information contained in voxel patterns for the analysis of functional connectivity. We achieve this by computing the mutual information of a pattern of locally connected voxels at the seed location with a pattern at the target location. As with the univariate functional connectivity analysis, this multivariate version also allows us to test hypotheses about connectivity of brain regions as well as generate connectivity maps.

Because of the large number of voxels in the brain (many thousands, depending on resolution), multivariate techniques usually require some kind of feature selection or dimensionality reduction. This can be achieved by focusing on pre-defined ROIs, or by selecting voxels form the brain based on some statistical criteria [3, 14]. Here we show that using mutual information of individual voxels with respect to ground truth for voxel selection works at the same level as these previous methods, but that mutual information of patterns of voxels with respect to ground truth outperforms all of the univariate methods we tested.

Information theory has been applied to fMRI data in the context of brain machine interfaces [25], to generate activation maps [8], for effective connectivity in patients [7], and for image registration [17]. However, to our knowledge this is the first application to both voxel selection and functional connectivity based on multivariate activity patterns.

## 4 Experiments

### 4.1 Data

For the experiments described in this section we use the data from the fMRI experiment on natural scene categories by [23]. Briefly, five participants passively viewed color images belonging to six categories of natural scenes (beaches, buildings, forests, highways, industry, and mountains). Stimuli were arranged into blocks of 10 images from the same natural scene category. Each image was displayed sequentially for 1.6 seconds. A run was composed of 6 blocks, one for each natural scene category, interleaved with 12 s fixation periods. Images were presented upright inverted on alternating runs, with each inverted run preserving the image and category order used in the preceding upright run. A session contained 12 such runs, and the order of categories was randomized across blocks. Each subject performed two blocks with a total of 24 runs. In total we have 1192 data points per subject across all 6 categories. The data obtained from the authors in [23] contains only localizers for V1, PPA, RSC, LOC, FFA areas. Thus we limit our seed areas to these ROIs.

### 4.2 Voxel selection

The goal of voxel selection is to identify the most relevant voxels for the experiment task out of the tens of thousands of voxels in the entire brain. A quantitative evaluation of voxel selection is the decoding accuracy[3] of the selected voxels, which measures how well can the selected voxels predict the viewing condition from the neural responses.

Fig.1 compares our mutual information-based voxel selection method to other voxel selection methods. Decoding accuracy[3] using univariate kNN mutual information is comparable to most active[1]

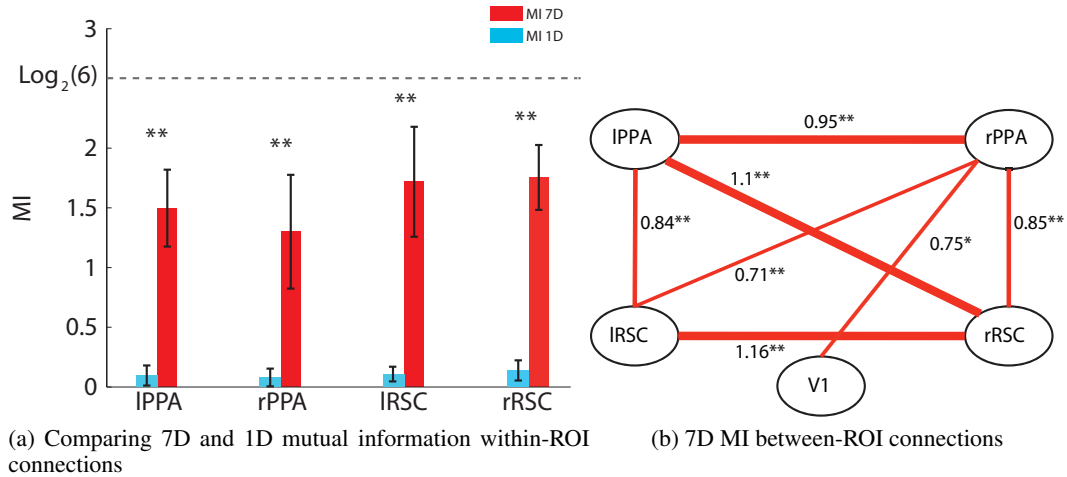

(a) Comparing 7D and 1D mutual information within-ROI connections

(b) 7D MI between-ROI connections

Figure 3: a) Within-ROI MI values for 7D and 1D mutual information, b) Schematic showing the significant ROI connections found using 7D mutual information analysis. The network shows strong connections between PPA and RSC, both ipsilaterally and contralaterally. $^{**}p < 10^{-6}$, $^{*}p < 0.01$

voxel selection. Multivariate information measure is able to select the more informative voxels by considering a local pattern of voxels jointly, leading to a boost in decoding accuracy[3].

To further understand why multivariate mutual information boosts the decoding accuracy[3], we can look at the spatial locations of the informative voxels selected by multivariate information analysis shown in Fig.2. The most informative voxels selected correspond to known functional regions for scenes. In this figure we see the scene areas V1, RSC, PPA, LOC that were also identified in [23]. Interestingly, our automatic voxel selection achieves a higher decoding accuracy[3] than the ROIs selected by localizer in [23]. This may suggest that the multivariate information voxel selection is a better segmentation of the relevant ROIs than the localizer runs.

## 4.3 Functional connectivity of ROIs

In the previous section, we have shown that multivariate information can effectively select informative voxels for classification. In this section, we first illustrate the increased sensitivity of a multivariate assessment of functional connectivity within known ROIs. Then we use multivariate information to explore connections between ROIs.

A good comparison for the functional connectivity measure is the within-ROI connectivity. Voxels within the same ROI should exhibit high functional connectivity with each other. In Fig.3a we compared our 7D measures with equivalent one dimensional measures using within-ROI connectivity. To this end we randomly selected 15 seed and 15 target locations within each ROI, making sure that seed and target patterns have no voxels in common. Then we computed mutual information between all seed and all target locations, either using individual voxels (1D case) or patterns of seven voxels (7D case). Fig.3a shows the mean of the mutual information values for these two cases in each ROI. In all ROIs, we find that multivariate information measure(7D) is significantly higher than the univariate measure(1D), suggesting that a pattern-based mutual information has a higher fidelity than univariate-based mutual information in mapping out functional connections.

After having established that 7D mutual information significantly outperforms 1D mutual information we proceed to calculate the between-ROI connectivity for scene areas V1, left/right PPA, and left/right RSC using 7D mutual information as shown in Fig.3b. Between-ROI connectivity is de-

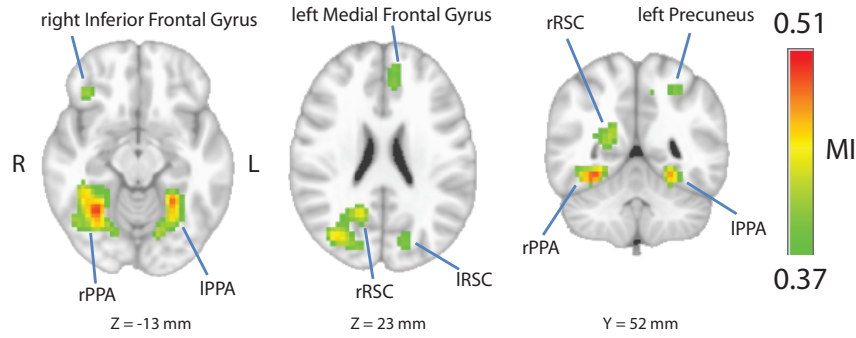

Figure 4: Connectivity map seeding from left PPA. Talairach coordinates defined in [22] are shown as the Z and Y coordinates for axial and coronal slices respectively. The intensity of the maps shows the MI values(This figure must be viewed in color)

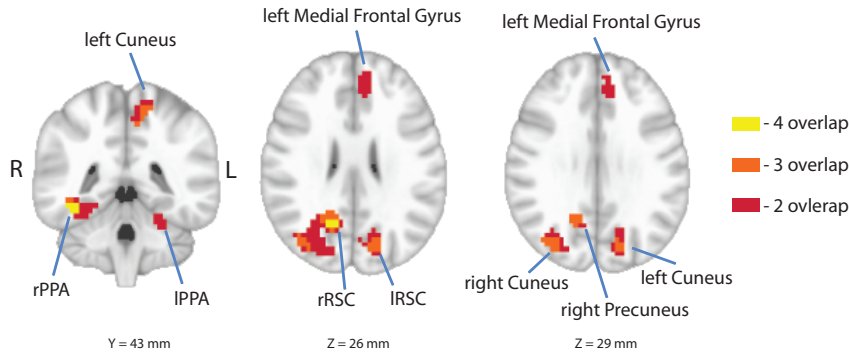

Figure 5: Overlap analysis showing areas where overlap occurs with the strongest connections from more than two scene network ROIs. Talairach coordinates defined in [22] are shown as the Z and Y coordinates for axial and coronal slices respectively. The color code indicates the amount of overlap.(This figure must be viewed in color)

fined similarly as within-ROI connectivity except that seed and target locations are the chosen in different ROIs.

A number of aspects of the connections mapped out with MI7D analysis agree with neuroscience findings. First, it is expected that PPA and RSC should be strongly connected as part of a scene network. Moreover, since V1 is the input to the cortical visual system, it is also likely that it should share information with at least one member of the scene network, which in this case was the right PPA. One novel finding from this analysis is that all of the strongest connections we discovered included right RSC. In particular, right RSC shares strong connections with left RSC, right PPA and left PPA, suggesting that right RSC may play a particularly important role in distinguishing natural scene categories. More work will be needed to verify this hypothesis.

To summarize, we have verified that multivariate information analysis can reliably map out connections within and between ROIs known to be involved in processing natural scene categories. In the next section we show how the same analysis can be extended to uncover other ROIs that share information with this scene network in our scene classification experiment.

### 4.4 Functional connectivity - whole brain analysis

While it is valuable to confirm existing hypotheses about areas that represent scene categories, it is also interesting to uncover new brain areas that might be related to scene categorization. In this section, we show that we can use our multivariate information analysis approach to explore other areas outside of the known ROIs that form strong connections with the known ROIs.

For each of the functional areas in the scene network, we can explore other areas connected to it. As in section 3 we measure functional connectivity as multivariate mutual information between the seed and candidate target areas. We fix the seed area to an ROI defined by a localizer. The candidate area moves around the brain, at each location measuring the mutual information with respect to the seed area.

#### 4.4.1 Confirming known connections

Fig.4 shows an example of the connectivity map seeding from left PPA. Each highlighted location in the connectivity map shows its connectivity to left PPA as measured by the multivariate information. As shown in Fig.4, both left and right PPA are highlighted, confirming their bilateral connection. Furthermore, we see strong connections between left PPA and left and right RSC. A minimum cluster size of 13 is used to threshold the connectivity map. The minimum cluster size is determined by AlphaSim in AFNI [4]. Notice in Fig.4 that the highest MI in the whole-brain analysis has MI of 0.51 whereas the within-ROI MI of left PPA in Fig.3b has a value of 1.5. The decrease in MI is due to the smoothing of connectivity maps when we combine them across subjects.

#### 4.4.2 Discovering new connections

Besides confirming known regions of the scene network, our connectivity maps allow us to explore other brain areas that might be related to the scene network. In Fig.4 we not only observe known scene network ROIs but additional areas such as the right Inferior Frontal Gyrus, left Medial Frontal Gyrus, and left Precuneus. Interestingly, the Inferior Frontal Gyrus, typically associated with language processing [13], also showed up in a searchlight analysis for decoding accuracy in [23].

So far we have examined how the rest of the brain connects to one ROI in the scene network, specifically we used left PPA as the example. However, to further strongly establish which regions are functionally connected in regards to distinguishing scene category, we asked which brain areas are strongly connected to two or more of the scene network ROIs. Areas that connect to more than one of the scene network ROIs are particularly interesting, because having multiple connections strengthens evidence that they play a significant role in distinguishing scene categories.

To investigate this question, we generate one connectivity map for each of the 4 scene network ROIs, similar to Fig.4. We take the areas with the top 5 percent highest mutual information in each of the 4 maps and overlap them. Fig.5 shows this overlap analysis.

Similar to the previous analysis, the overlap analysis highlights all 4 known areas of the scene network. Interestingly, this analysis shows that right RSC and right PPA are connected with more regions of the scene network than left RSC and PPA. This suggests that perhaps there is a laterality effect in the scene network that could be investigated in future studies.

Furthermore, we can also explore areas outside of the scene network with the overlap analysis. In Fig.5, left/right Cuneus and right Precuneus, highlighted in orange, exhibit strong connections with 3/4 of the scene network ROIs. Left Medial Frontal Gyrus is strongly connected to 2/4 of the scene network ROIs. These exploratory areas also point to interesting future investigations for scene category studies.

## 5 Conclusion

In this paper we have introduced a new method for evaluating the mutual information that patterns of fMRI voxels share with the ground truth labels of the experiment and with patterns of voxels elsewhere in the brain. When used as a voxel selection method for subsequent decoding of viewed natural scene category, mutual information of patterns of voxels with respect to the ground truth label is superior to mutual information of individual voxels.

We have shown that mutual information of voxel patterns in two ROIs is a more sensitive measure of task-specific functional connectivity analysis than mutual information of individual voxels. We have identified a network of regions consisting of left and right PPA and left and right RSC that share information about the category of a natural scene viewed by the subject. Connectivity maps generated with this method have identified left medial frontal gyrus, left/right cuenus, and right precuneus as sharing scene-specific information with PPA and RSC. This could stimulate interesting future work such as estimating mutual information for an even larger set of voxels and understanding the exploratory areas highlighted by this analysis. Although we confined our experiments to data from a scene category task, all the analysis proposed here could be used for other tasks in other domains.

**Acknoledgements**

This work is funded by National Institutes of Health Grant 1 R01 EY019429 (to L.F.-F., D.M.B., D.B.W.), a Beckman Postdoctoral Fellowship (to D.B.W.), a Microsoft Research New Faculty Fellowship (to L.F.-F.), and the Frank Moss Gift Fund (to L.F-F.). The authors would like to thank Todd Coleman and Fernando Perez-Cruz for the helpful discussions on entropy estimation.

## Footnotes

*Barry Chai and Dirk B. Walther contributed equally to this work.

†Diane M. Beck and Li Fei-Fei contributed equally to this work.

[1]Most discri. – Most discriminative voxels are those showing the largest difference in activity between any pair of scene categories.

[2]Most active – Most active voxels are those showing the largest difference in activity between the fixation condition and viewing images of any category.

[3]Decoding accuracy is obtained with a leave-two-runs-out cross-validation on the our scene data. In each fold two runs from viewing the same images upright and inverted are left out as test data. Voxel selection is performed on the training runs using $k = n/2$, where $n$ is the number of training examples in each category. Using selected voxels, a linear SVM classifier is trained on the upright runs with $C = 0.02$ as in [23]. In testing we use majority voting on the SVM prediction labels to vote for the most likely scene label for each block of data. Decoding accuracy is the average of cross-validation accuracy over the 5 subjects.

# References

[1] Granger, C. W. J. Investigating causal relations by econometric models and cross-spectral methods. *Econometrica 37, 424-438, 1969*

[2] J. Cao and K. Worsley The geometry of correlation fields with an application to functional connectivity of the brain. *Ann. Appl. Probab*, 9:1021C1057, 1998.

[3] D. Cox and R. Savoy Functional magnetic resonance imaging (fMRI) "brain reading": Detecting and classifying distributed patterns of fMRI activity in human visual cortex. *NeuroImage*, 19(2):261C270, 2003.

[4] RW Cox. AFNI: Software for analysis and visualization of functional magnetic resonance neuroimages. *Computers and Biomedical Research, 29:162-173, 1996.*

[5] K. J. Friston, C. D. Frith, P. F. Liddle, and R. S. Frackowiak. Functional connectivity: the principal-component analysis of large (PET) data sets. *J Cereb Blood Flow Metab*, 13(1):5C14, January 1993.

[6] J. V. Haxby, M. I. Gobbini, M. L. Furey, A. Ishai, J. L. Schouten, and P. Pietrini. Distributed and overlapping representations of faces and objects in ventral temporal cortex. *Science*, 293(5539):2425C30, 2001. Journal Article United States.

[7] H. Hinrichs, H. Heinze, and M. Schoenfeld. Causal visual interactions as revealed by an information theoretic measure and fMRI. *NeuroImage*, 31(3):1051 C 1060, 2006.

[8] A. T. John, J. W. F. Iii, W. M. W. Iii, J. Kim, and A. S.'Willsky. Analysis of functional MRI data using mutual information, 1999.

[9] Fei-Fei, L., Iyer, A., Koch, C., Perona, P. (2007). What do we perceive in a glance of a real-world scene? *Journal of Vision,7(1):10, 1-29, http://journalofvision.org/7/1/10/,doi:10.1167/7.1.10.*

[10] K. Li, L. Guo, J. Nie, G. Li, and T. Liu. Review of methods for functional brain connectivity detection using fMRI. *Computerized medical imaging and graphics: The Official Journal of the Computerized Medical Imaging Society*, 33(2):131C139, March 2009.

[11] M. J. Mckeown, S. Makeig, G. G. Brown, T.-P. Jung, S. S. Kindermann, R. S. Kindermann, A. J. Bell, and T. J. Sejnowski. Analysis of fMRI data by blind separation into independent spatial components. *Human Brain Mapping*, 6:160C188, 1998.

[12] A. Meyer-Baese, A. Wismueller, and O. Lange. Comparison of two exploratory data analysis methods for fMRI: unsupervised clustering versus independent component analysis. *Information Technology in Biomedicine*, IEEE Transactions on, 8(3):387C398, Sept. 2004.

[13] Geschwind N. (1970) The organization of language and the brain. *Science 170:940 944..*

[14] D. Neill, A. Moore, F. Pereira, and T. Mitchell. Detecting significant multidimensional spatial clusters. *In Proceedings of Neural Information Processing Systems*, 2004.

[15] M. Potter. Short-term conceptual memory for pictures. *Journal of Experimental Psychology: Human Learning and Memory*, 2(5):509C522, 1976.

[16] F. Perez-Cruz. Estimation of information theoretic measures for continuous random variables. *In D. Koller, D. Schuurmans, Y. Bengio, and L. Bottou, editors, NIPS*, pages 1257C1264. MIT Press, 2008.

[17] Pluim, J. P. W. and Maintz, J. B. A. and Viergever, M. A. Mutual-information based registration of medical images: a survey. *IEEE Trans Med Imaging*, 2003, 22:986C1004.

[18] X. U. Rui, C. H. E. N. Yen-Wei, T. A. N. G. Song-Yuan, S. Morikawa, and Y. Kurumi. Parzen-window based normalized mutual information for medical image registration. *IEICE Transactions on Information and Systems*, E91-D(1):132C144, January 2008.

[19] C. E. Shannon. A Mathematical Theory of Communication. *CSLI Publications*, 1948.

[20] T. Schreiber. Measuring Information Transfer. *Physical Review Letters*, vol. 85, no. 2, pp. 461+, July 2000.

[21] F. T. Sun, L. M. Miller, and M. DEsposito. Measuring interregional functional connectivity using coherence and partial coherence analyses of fmri data. *NeuroImage*, 21(2):647 C 658, 2004.

[22] J. Talairach and P. Tournoux. Co-planar Stereotaxic Atlas of the Human Brain: 3-Dimensional Proportional System - an Approach to Cerebral Imaging. *Thieme Medical Publishers*, New York, 1988

[23] D. Walther, E. Caddigan, L. Fei-Fei*, and D. Beck* (2009), Natural scene categories revealed in distributed patterns of activity in the human brain. *The Journal of Neuroscience*, 29(34):10573-10581.(*indicates equal contribution)

[24] Q. Wang, S. Kulkarni, and S. Verdu. Divergence estimation of continuous distributions based on data-dependent partitions. *Information Theory, IEEE Transactions on*, 51(9):3064C3074, Sept. 2005.

[25] B. D. Ward and Y. Mazaheri. Information transfer rate in fMRI experiments measured using mutual information theory. *Journal of Neuroscience Methods*, 167(1):22 C 30, 2008. Brain-Computer Interfaces (BCIs).

[26] B. Yao, D.B. Walther, D.M. Beck*, L. Fei-Fei*. Hierarchical Mixture of Classification Experts Uncovers Interactions between Brain Regions. *NIPS*, 2009. (* indicates equal contribution)

